# A SELF-LEARNING NEURAL NETWORK

A. Hartstein and R. H. Koch
IBM - Thomas J. Watson Research Center
Yorktown Heights, New York

## ABSTRACT

We propose a new neural network structure that is compatible with silicon technology and has built-in learning capability. The thrust of this network work is a new synapse function. The synapses have the feature that the learning parameter is embodied in the thresholds of MOSFET devices and is local in character. The network is shown to be capable of learning by example as well as exhibiting the desirable features of the Hopfield type networks.

The thrust of what we want to discuss is a new synapse function for an artificial neuron to be used in a neural network. We choose the synapse function to be readily implementable in VLSI technology, rather than choosing a function which is either our best guess for the function used by real synapses or mathematically the most tractable. In order to demonstrate that this type of synapse function provides interesting behavior in a neural network, we imbed this type of function in a Hopfield {Hopfield, 1982} type network and provide the synapses with a Hebbian {Hebb, 1949} learning capability. We then show that this type of network functions in much the same way as a Hopfield network and also learns by example. Some of this work has been discussed previously {Hartstein, 1988}.

Most neural networks, which have been described, use a multiplicative function for the synapses. The inputs to the neuron are multiplied by weighting factors and then the results are summed in the neuron. The result of the sum is then put into a hard threshold device or a device with a sigmoid output. This is not the easiest function for a MOSFET to perform although it can be done. Over a large range of parameters, a MOSFET is a linear device with the output current being a linear function of the input voltage relative to a threshold voltage. If one could directly utilize these characteristics, one would be able to design a neural network more compactly.

We propose that we directly use MOSFETs as the input devices for the neurons in the network, utilizing their natural characteristics. We assume the following form for the input of each neuron in our network:

$$V_i = \sigma \left( \sum_j |V_j - T_{ij}| \right) \tag{1}$$

where $V_i$ is the output, $V_j$ are the inputs and $T_{ij}$ are the learned threshold voltages. In this network we use a representation in which both the V's and the T's range from 0 to +1. The result of the summation is fed into a non-linear sigmoid function ($\sigma$). All of the neurons in the network are interconnected, the outputs of each neuron feeding the inputs of every other neuron. The functional form of Eq. 1 might, for instance, represent several n-channel and p-channel MOSFETs in parallel.

The memories in this network are contained in the threshold voltages, $T_{ij}$. We implement learning in this network using a simple linear Hebbian {Hebb, 1949} learning rule. We use a rule which locally reinforces the state of each input node in a neuron relative to the output of that neuron. The equation governing this learning algorithm is:

$$T'_{ij} = T_{ij} + \eta(|V_j - V_i| - 0.5) \tag{2}$$

where $T_{ij}$ are the initial threshold voltages and $T'_{ij}$ are the new threshold voltages after a time, $\Delta t$. Here $\eta$ is a small learning parameter related to this time period, and the offset factor 0.5 is needed for symmetry. Additional saturation constraints are imposed to ensure that $T_{ij}$ remain in the interval 0 to +1.

This learning rule is one which is linear in the difference between each input and output of a neuron. This is an enhancing/inhibiting rule. The thresholds are adjusted in such a way that the output of the neuron is either pushed in the same direction as the input (enhancing), or pushed in the opposite direction (inhibiting). For our simple simulations we started the network with all thresholds at 0.5 and let learning proceed until some saturation occurred. The somewhat more sophisticated method of including a relaxation term in Eq. 2 to slowly push the values toward 0.5 over time was also explored. The results are essentially the same as for our simple simulations.

The interesting question is if we form a network using this type of neuron, what will the overall network response be like? Will the network learn multiple states or will it learn a simple average over all of the states it sees? In order to probe the functioning of this network, we have performed simulations of this network on a digital computer. Each simulation was divided into two phases. The first was a learning phase in which a fixed number of random patterns were presented to the network sequentially for some period of time. During this phase the threshold

voltages were allowed to change using the rule in Eq. 2. The second was a testing phase in which learning was turned off and the memories established in the network were probed to determine the essential features of these learned memories. In this way we could test how well the network was able to learn the initial test patterns, how well the network could reconstruct the learned patterns when presented with test patterns containing errors, and how the network responded to random input patterns.

We have simulated this network using N fully interconnected neurons, with N in the range of 10 to 200. M random patterns were chosen and sequentially presented to the network for learning. M typically ranged up to N/3. After the learning phase, the nature of the stable states in the network was tested. In general we found that the network is capable of learning all of the input patterns as long as M is not too large. The network also learns the inverse patterns (1's and 0's interchanged) due to the inherent symmetry of the network. Additional extraneous patterns are learned which have no obvious connection to the intended learned states. These may be analogous to either the spin glass states or the mixed pattern states discussed for the multiplicative network {Amit, 1985}.

Fig. 1 shows the capacity of a 100 neuron network. We attempted to teach the network M states and then probed the network to see how many of the states were successfully learned. This process was repeated many times until we achieved good statistics. We have defined successful learning as 100% accuracy. A more relaxed definition would yield a qualitatively similar curve with larger capacity.

The functional form of the learning is peaked at a fixed value of the number of input patterns. For a small number of input patterns, the network essentially learns all of the patterns. Deviations from perfect learning here generally mean 1 bit of information was learned incorrectly. Near the peak the results become more noisy for different learning attempts. Most errors are still only 1 or 2 bits, but the learning in this region becomes marginal as the capacity of the network is approached. For larger values of the number of input patterns the network becomes overloaded and it becomes incapable of learning most of the input states. Some small number of patterns are still learned, but the network is clearly not functioning well. Many of the errors in this region are large, showing little correlation with the intended learned states.

This functional form for the learning in the network is the same for all of the network sizes tested. We define the capacity of the network as the average value of the peak number of patterns which can be successfully learned. The inset to Fig. 1 shows the memory capacity of a number of tested networks as a function of the size of the network. The network capacity is seen to be a linear function of the network size. The capacity is proportional to the number of $T_{ij}$'s specified. In this

example the network capacity was found to be about 8% of the maximum possible for binary information. This rather low figure results from a trade-off of capacity for the particular types of functions that a neural network can perform. It is possible to construct simple memories with 100% capacity.

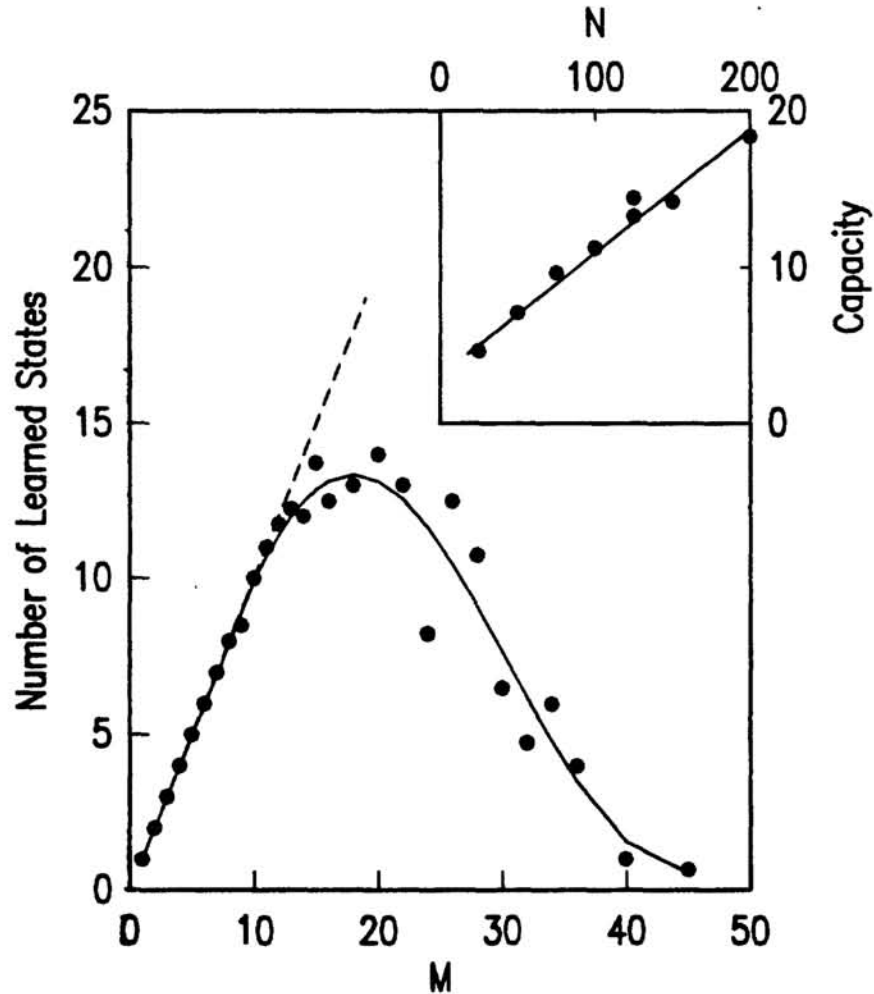

**Figure 1.** The number of successfully learned patterns as a function of the number of input patterns for a 100 neuron network. The dashed curve is for perfect learning. The inset shows the memory capacity of a threshold neural network as a function of the size of the network.

Some important measures of learning in the network are the distribution of stable states in the network after learning has taken place, and the basin of attraction for each stable point. One can gain a handle on these parameters by probing the network with random test patterns after the network has learned M states. Fig. 2 shows the averaged results of such tests for a 100 neuron network and varying numbers of learned states. The figure shows the probability of finding particular states, both learned and extraneous. The states are ordered first by decreasing

probability for the learned states, followed by decreasing probability for the extraneous states. It is clear from the figure that both types of stable states are present in the network. It is also clear that the probabilities of finding different patterns are not equal. Some learned states are more robust than others, that is they have larger basins of attraction. This network model does not partition the available memory space equally among the input patterns. It also provides a large amount of memory space for the extraneous states. Clearly, this is not the optimum situation.

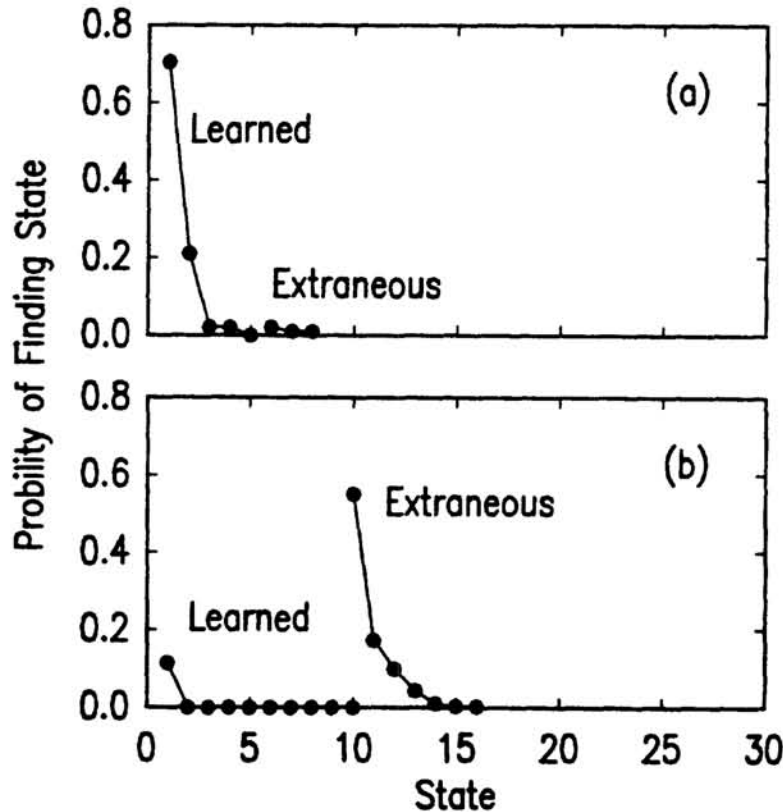

**Figure 2.** The probability of the network finding a specific pattern. Both learned states and extraneous states are found. The figure was obtained for a 100 neuron network. Fig. 2a is for 5 learned patterns and 2b is for 10 learned patterns.

Some of the learned states appear to have 0 probability of being found in this simulation. Some of these states are not stable states of the network and will never be found. This is particularly true when the number of learned states is close to or exceeds the capacity of the network. Others of these states simply have an extremely small probability of being found in a random search because they have small basins of attraction. However, as discussed below, these are still viable states. When the network learns fewer states than its capacity (Fig. 2a),

most of the stable states are the learned states.  As the capacity is approached or exceeded, most of the stable states are extraneous states.

The results shown in Fig. 2 address the question of the networks tolerance to errors.  A pattern, which has a large basin of attraction, will be relatively tolerant to errors when being retrieved, whereas, a pattern, which has a small basin of attraction, will be less tolerant of errors.  The immunity of the learned patterns to errors in being retrieved can also be tested in a more direct way.  One can probe the network with test patterns which start out as the learned patterns, but have a certain number of bits changed randomly.  One then monitors the final pattern which the networks finds and compares to the known learned pattern.

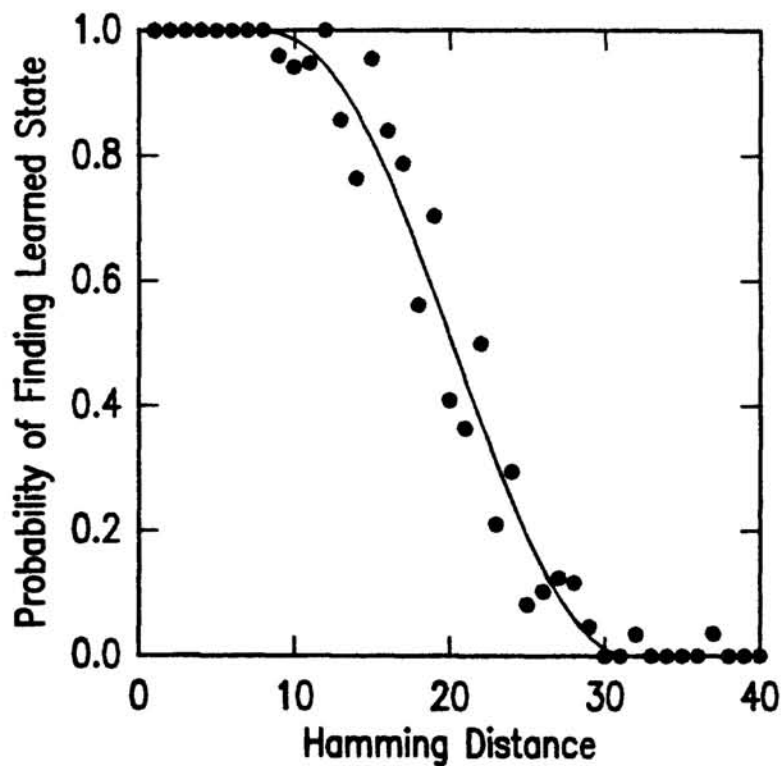

Figure 3.  Probability of the network finding a specific learned state when the input pattern has a certain Hamming distance.  This figure was obtained for a 100 neuron network which was taught 10 random patterns.

Fig. 3 shows typical results of such a calculation.  The probability of successfully retrieving a pattern is shown as a function of the Hamming distance, the number of bits which were randomly changed in the test pattern.  For this simulation a 100 neuron network was used and it was taught 10 patterns.  For small Hamming distances the patterns are successfully found 100% of the time.  As the Hamming distance gets larger the network is no longer capable of finding the desired pattern, but rather finds one of the other fixed points.  This result is a statistical av-

erage over all of the states and therefore tends to emphasize patterns with small basins of attraction. This is just the opposite of the types of states emphasized in the analysis shown in Fig. 2.

We can define the maximum Hamming distance as the Hamming distance at which the probability of finding the learned state has dropped to 50%. Fig. 4 shows the maximum Hamming distance as a function of the number of learned states in our 100 neuron network. As one expects the maximum Hamming distance gets smaller as the number of learned states increases. Perhaps surprisingly, the relationship is linear. These results are important since one requires a reasonable maximum Hamming distance for any real system to function. These considerations also shed some light on the nature of the functioning of the network and its ability to learn.

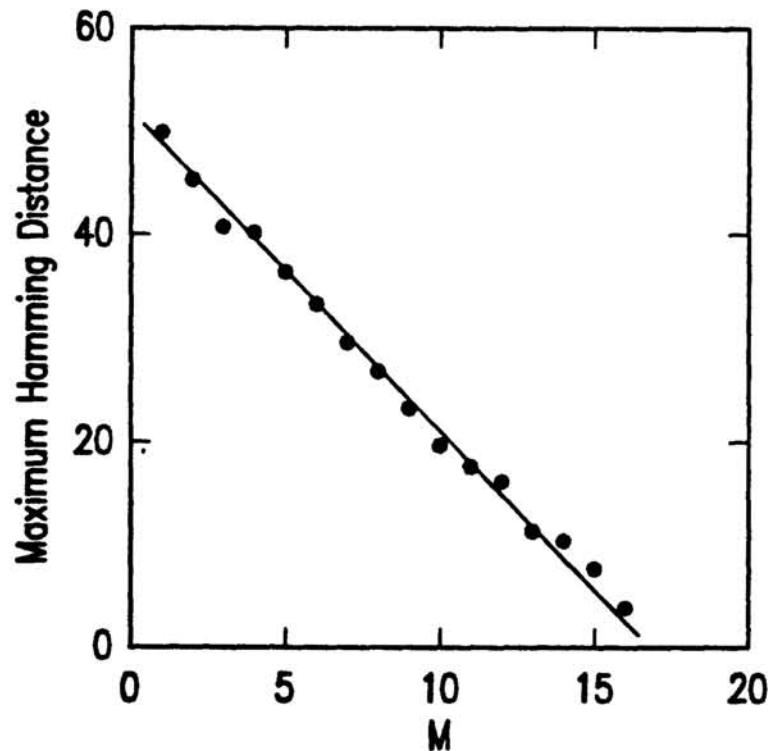

Figure 4. The maximum Hamming distance for a given number of learned states. Results are for a 100 neuron network.

This simulation gives us a picture of the way in which the network utilizes its phase space to store information. When only a few patterns are stored in the network, the network divides up the available space among these memories. The learning process is almost always successful. When a larger number of learned patterns are attempted, the available space is now divided among more memories. The maximum Hamming distance decreases and more space is taken up by extraneous states. When the memory capacity is exceeded, the phase space allo-

cated to any successful memory is very small and most of the space is taken up by extraneous states.

The types of behavior we have described are similar to those found in the Hopfield type memory utilizing multiplicative synapses. In fact our central point is that by using a completely different type of synapse function, we can obtain the same behavior. At the same time we argue since this network was proposed using a synapse function which mirrors the operating characteristics of MOSFETs, it will be much easier to realize in hardware. Therefore, we should be able to construct a smaller more tolerant network with the same operating characteristics.

We do not mean to imply that the type of synapse function we have explored can only be used in a Hopfield type network. In fact we feel that this type of neuron is quite general and can successfully be utilized in any type of network. This is at present just a conjecture which needs to be explored more fully. Perhaps the most important message from our work is the realization that one need not be constrained to the multiplicative type of synapse, and that other forms of synapses can perform similar functions in neural networks. This may open up many new avenues of investigation.

## REFERENCES

D.J. Amit, H. Gutfreund and H. Sompolinsky, Phys. Rev. A32, 1007 (1985).

A. Hartstein and R.H. Koch, IEEE Int. Conf. on Neural Networks, (SOS Printing, San Diego, 1988), Vol. I, 425.

D. Hebb, The Organization of Behaviour, (Wiley, New York, 1949).

J.J. Hopfield, Proc. Natl. Acad. Sci. USA 79, 2554 (1982).
